# LEARNING IN NETWORKS OF
# NONDETERMINISTIC ADAPTIVE LOGIC ELEMENTS

Richard C. Windecker*
AT&T Bell Laboratories, Middletown, NJ  07748

## ABSTRACT

This paper presents a model of nondeterministic adaptive automata that are constructed from simpler nondeterministic adaptive information processing elements. The first half of the paper describes the model. The second half discusses some of its significant adaptive properties using computer simulation examples. Chief among these properties is that network aggregates of the model elements can adapt appropriately when a *single* reinforcement channel provides the same positive or negative reinforcement signal to *all* adaptive elements of the network at the same time. This holds for multiple-input, multiple-output, multiple-layered, combinational and sequential networks. It also holds when some network elements are "hidden" in that their outputs are not directly seen by the external environment.

## INTRODUCTION

There are two primary motivations for studying models of adaptive automata constructed from simple parts. First, they let us learn things about real biological systems whose properties are difficult to study directly: We form a hypothesis about such systems, embody it in a model, and then see if the model has reasonable learning and behavioral properties. In the present work, the hypothesis being tested is: that much of an animal's behavior as determined by its nervous system is intrinsically *non*deterministic; that learning consists of *incremental* changes in the probabilities governing the animal's behavior; and that this is a consequence of the animal's nervous system consisting of an aggregate of information processing elements some of which are individually *non*deterministic and adaptive. The second motivation for studying models of this type is to find ways of building machines that can *learn* to do (artificially) intelligent and practical things. This approach has the potential of complementing the currently more developed approach of programming intelligence into machines.

We do not assert that there is necessarily a one-to-one correspondence between real physiological neurons and the postulated model information processing elements. Thus, the model may be loosely termed a "neural network model," but is more accurately described as a model of adaptive automata constructed from simple adaptive parts.

---

* The main ideas in this paper were conceived and initially developed while the author was at the University of Chiang Mai, Thailand (1972-73). The ideas were developed further and put in a form consistent with existing switching and automata theory during the next four years. For two of those years, the author was at the University of Guelph, Ontario, supported of National Research Council of Canada Grant #A6983.

It almost certainly has to be a property of any acceptable model of animal learning that a *single* reinforcement channel providing reinforcement to *all* the adaptive elements in a network (or subnetwork) can effectively cause that network to adapt appropriately. Otherwise, methods of providing separate, specific reinforcement to all adaptive elements in the network must be postulated. Clearly, the environment reinforces an animal as a whole and the *same* reinforcement mechanism can cause the animal to adapt to *many* types of situation. Thus, the reinforcement system is non-specific to particular adaptive elements and particular behaviors. The model presented here has this property.

The model described here is a close cousin to the family of models recently described by Barto and coworkers [1-4]. The most significant difference are: 1) In the present model, we define the timing discipline for networks of elements more explicitly and completely. This particular timing discipline makes the present model consistent with a nondeterministic extension of switching and automata theory previously described [5]. 2) In the present model, the reinforcement algorithm that adjusts the weights is kept very simple. With this algorithm, positive and negative reinforcement have symmetric and opposite effects on the weights. This ensures that the logical signals are symmetric opposites of each other. (Even small differences in the reinforcement algorithm can make both subtle as well as profound differences in the behavior of the model.) We also allow, null, or zero, reinforcement.

As in the family of models described by Barto, networks constructed within the present model can get "stuck" at a suboptimal behavior during learning and therefore not arrive at the optimal adapted state. The complexity of the Barto reinforcement algorithm is designed partly to overcome this tendency. In the present work, we emphasize the use of *training strategies* when we wish to ensure that the network arrives at an optimal state. (In nature, it seems likely that getting "stuck" at suboptimal behavior is common.) In all networks studied so far, it has been easy to find strategies that prevent the network from getting stuck.

The chief contributions of the present work are: 1) The establishment of a close connection between these types of models and ordinary, nonadaptive, switching and automata theory [5]. This makes the wealth of knowledge in this area, especially network synthesis and analysis methods, readily applicable to the study of adaptive networks. 2) The experimental demonstration that *sequential* ("recurrent") nondeterministic adaptive networks can adapt appropriately. Such networks can learn to produce outputs that depend on the recent *sequence* of past inputs, not just the current inputs. 3) The demonstration that the use of *training strategies* can not only prevent a network from getting stuck, but may also result in more rapid learning. Thus, such strategies may be able to compensate, or even more than compensate, for reduced complexity in the model itself.

References 2-4 and 6 provide a comprehensive background and guide to the literature on both deterministic and nondeterministic adaptive automata including those constructed from simple parts and those not.

## THE MODEL ADAPTIVE ELEMENT

The model adaptive element postulated in this work is a nondeterministic, adaptive generalization of threshold logic [7]. Thus, we call these elements Nondeterministic Adaptive Threshold-logic gates (NATs). The output chosen by a NAT at any given time is *not* a function of its inputs. Rather, it is chosen by a stochastic process according to certain probabilities. It is these *probabilities* that are a function of the inputs.

A NAT is like an ordinary logic gate in that it accepts logical inputs that are two-valued and produces a logical output that is two-valued. We let these values be

+1 and −1. A NAT also has a timing input channel and a reinforcement input channel. The NAT operates on a three-part cycle: 1) Logical input signals are changed and remain constant. 2) A timing signal is received and the NAT selects a new output based on the inputs at that moment. The new output remains constant. 3) A reinforcement signal is received and the weights are incremented according to certain rules.

Let $N$ be the number of logical input channels, let $x_i$ represent the $i^{th}$ input signal, and let $z$ be the output. The NAT has within it $N+1$ "weights," $w_0, w_1, ..., w_N$. The weights are confined to integer values. For a given set of inputs, the gate calculates the quantity $W$:

$$W = w_0 + w_1 x_1 + w_2 x_2 + w_3 x_3 + .... + w_N x_N = w_0 + \vec{W} \cdot \vec{X} \qquad (1)$$

Then the probability that output $z = +1$ is chosen is:

$$P(z = +1) = \frac{1}{\sqrt{2\pi}\sigma} \int_{-\infty}^{W} e^{-\frac{x^2}{2\sigma^2}} \, dx = \frac{1}{\sqrt{\pi}} \int_{-\infty}^{W/\sqrt{2}\sigma} e^{-\varsigma^2} \, d\varsigma \qquad (2)$$

where $\varsigma = x/\sqrt{2}\sigma$. (An equivalent formulation is to let the NAT generate a random number, $w_\sigma$, according to the normal distribution with mean zero and variance $\sigma^2$. Then if $W > -w_\sigma$, the gate selects the output $z = +1$. If $W < -w_\sigma$, the gate selects output $z = -1$. If $W = -w_\sigma$, the gate selects output −1 or +1 with equal probability.)

Reinforcement signals, $R$, may have one of *three* values: +1, −1, and 0 representing positive, negative, and no reinforcement, respectively. If +1 reinforcement is received, each weight is incremented by one in the direction that makes the current output, $z$, *more likely* to occur in the future when the same inputs are applied; if −1 reinforcement is received, each weight is incremented in the direction that makes the current output *less likely*; if 0 reinforcement is received, the weights are not changed. These rules may be summarized: $\Delta w_0 = zR$ and $\Delta w_i = x_i zR$ for $i > 0$.

NATs operate in discrete time because if the NAT can choose output +1 or −1, depending on a stochastic process, it has to be told *when* to select a new output. It cannot "run freely," or it could be constantly changing output. Nor can it change output only when its inputs change because it may need to select a new output even when they do not change.

The normal distribution is used for heuristic reasons. If a real neuron (or an aggregate of neurons) uses a stochastic process to produce nondeterministic behavior, it is likely that process can be described by the normal distribution. In any case, the *exact* relationship between $P(z = +1)$ and $W$ is not critical. What *is* important is that $P(z = +1)$ be monotonically increasing in $W$, go to 0 and 1 asymptotically as $W$ goes to $-\infty$ and $+\infty$, respectively, and equal 0.5 at $W = 0$.

The parameter $\sigma$ is adjustable. We use 10 in the computer simulation experiments described below. Experimentally, values near 10 work reasonably well for networks of NATs having few inputs. Note that as $\sigma$ goes to zero, the behavior of a NAT approximates that of an ordinary deterministic adaptive threshold logic gate with the difference that the output for the case $W = 0$ is not arbitrary: The NAT will select output +1 or −1 with equal probability.

Note that for *all* values of W, the probabilities are greater than zero that either +1 or −1 will be chosen, although for large values of W (relative to $\sigma$) for all

practical purposes, the behavior is deterministic. There are many values of the weights that cause the NAT to *approximate* the behavior of a deterministic threshold logic gate. For the same reasons that deterministic threshold logic gates cannot realize all $2^{2^N}$ functions of $N$ variables [7], so a NAT cannot learn to approximate *any* deterministic function; only the threshold logic functions.

Note also that when the weights are near zero, a NAT adapts most rapidly when both positive and negative reinforcement are used in approximately equal amounts. As the NAT becomes more likely to produce the appropriate behavior, the opportunity to use negative reinforcement decreases while the opportunity to use positive reinforcement increases. This means that a NAT cannot learn to (nearly) always select a certain output if negative reinforcement alone is used. Thus, positive reinforcement has an important role in this model. (In most deterministic models, positive reinforcement is not useful.)

Note further that there is no hysteresis in NAT learning. For a given configuration of inputs, a $+1$ output followed by a $+1$ reinforcement has exactly the same effect on all the weights as a $-1$ output followed by a $-1$ reinforcement. So the *order* of such events has no effect on the final values of the weights.

Finally, if only *negative* reinforcement is applied to a NAT, independent of output, for a particular combination of inputs, the weights will change in the direction that makes $W$ tend toward zero and once there, follow a random walk centered on zero. (The further $W$ is from zero, the more likely its next step will be toward zero.) If all possible input combinations are applied with more or less equal probability, *all* the weights will tend toward zero and then follow random walks centered on zero. In this case, the NAT will select $+1$ or $-1$ with more or less equal probability without regard to its inputs.

## NETWORKS

NATs may be connected together in networks (NAT-nets). The inputs to a NAT in such a network can be selected from among: 1) the set of inputs to the entire network, 2) the set of outputs from other NATs in the network, and 3) its own output. The outputs of the network may be chosen from among: 1) the inputs to the network as a whole, and 2) the outputs of the various NATs in the network.

Following Ref. 5, we impose a timing discipline on a NAT-net. The network is organized into layers such that each NAT belongs to one layer. Letting $L$ be the number of layers, the network operates as follows: 1) All NATs in a given layer receive timing signals at the same time and select a new output at the same time. 2) Timing signals are received by the different layers, in sequence, from 1 to $L$. 3) Inputs to the network as a whole are levels that may change only *before* Layer 1 receives its timing signal. Similarly, outputs from the network as a whole are available to the environment only *after* Layer $L$ has received its timing signal. Reinforcement to the network as a whole is accepted only *after* outputs are made available to the environment. The same reinforcement signal is distributed to *all* NATs in the network at the same time.

With these rules, NAT-nets operate through a sequence of timing cycles. In each cycle: 1) Network inputs are changed. 2) Layers 1 through $L$ select new outputs, in sequence. 3) Network outputs are made available to the environment. 4) Reinforcement is received from the environment. We call each such cycle a "trial" and a sequence of such trials is a "session."

This model is very general. If, for each gate, inputs are selected only from among the inputs to the network as a whole and from the outputs of gates in layers preceding it in the timing cycle, then the network is *combinational*. In this case, the probability of the network producing a given output configuration is a function of the inputs at the start of the timing cycle. If at least one NAT has one input from a

NAT in the same layer or from a subsequent layer in the timing cycle, then the network is *sequential*. In this case, the network may have "internal states" that allow it to remember information from one cycle to the next. Thus, the probabilities governing its choice of outputs may depend on inputs in previous cycles. So sequential NAT-nets may have short-term memory embodied in internal states and long-term memory embodied in the weights. In Ref. 5, we showed that sequential networks can be constructed by adding feedback paths to combinational networks and any sequential network can be put in this standard form.

In information-theoretic terms: 1) A NAT-net with no inputs and some outputs is an "information source." 2) A NAT-net with both inputs and outputs is an information "channel." 3) A combinational NAT-net is "memory-less" while a sequential NAT-net has memory. In this context, note that a NAT-net may operate in an environment that is either deterministic or nondeterministic. Both the logical and the reinforcement inputs can be selected by stochastic processes. Note also that nondeterministic and deterministic elements as well as adaptive and nonadaptive elements can be combined in one network. (It may be that the decision-making parts of an animal's nervous system are nondeterministic and adaptive while the information transmitting parts (sensory data-gathering and the motor output parts) are deterministic and nonadaptive.)

One capability that combinational NAT-nets possess is that of "pattern recognizers." A network having many inputs and one or a few outputs can "recognize" a small subset of the potential input patterns by producing a particular output pattern with high probability when a member of the recognized subset appears and a different output pattern otherwise. In practice, the number of possible input patterns may be so large that we cannot present them all for training purposes and must be content to train the network to recognize one subset by distinguishing it (with different output pattern) from another subset. In this case, if a pattern is subsequently presented to the network that has not been in one of the training sets, the probabilities governing its output may approach one or zero, but may well be closer to 0.5. The exact values will depend on the details of the training period. If the new pattern is similar to those in one of the training sets, the NAT-net will often have a high probability of producing the same output as for that set. This *associative* property is the analog of the well known associative property in deterministic models. If the network lacks sufficient complexity for the separation we wish to make, then it cannot be trained. For example, a single $N$-input NAT cannot be trained to recognize *any* arbitrary set of input patterns by selecting the $+1$ output when one of them is presented and $-1$ otherwise. It can only be trained to make separations that correspond to threshold functions.

A combinational NAT-net can also produce patterns. By analogy with a pattern recognizer, a NAT-net with none or a few inputs and a larger number of outputs can learn for each input pattern to produce a particular subset of the possible output patterns. Since the mapping may be few-to-many, instead of many-to-few, the goal of training in this case may or may not be to have the network approximate deterministic behavior. Clearly, the distinction between pattern recognizers and pattern producers is somewhat arbitrary: in general, NAT-nets are pattern transducers that map subsets of input patterns into subsets of output patterns. A *sequential* network can "recognize" patterns in the time-sequence of network inputs and produce patterns in the time-sequence of outputs.

SIMULATION EXPERIMENTS

In this Section, we discuss computer simulation results for three types of multiple-element networks. For two of these types, certain strategies are used to train the networks. In general, these strategies have two parts that alternate, as

needed. The first part is a general scheme for providing network inputs and reinforcement that tends to train all elements in the network in the desired direction. The second part is substituted temporarily when it becomes apparent that the network is getting stuck in some suboptimal behavior. It is focussed on getting the network unstuck. The strategies used here are intuitive. In general, there appear to be many strategies that will lead the network to the desired behavior. While we have made some attempt to find strategies that are reasonably efficient, it is very unlikely that the ones used are optimal. Finally, these strategies have been tested in hundreds of training sessions. Although they worked in all such sessions, there may be some (depending on the sequence of random numbers generated) in which they would not work.

In describing the networks simulated, Figs. 1-3, we use the diagramatic conventions defined in Ref. 5: We put all NATs in the same layer in a vertical line, with the various layers arranged from left to right in their order in the timing cycle. Inputs to the entire network come in from the left; outputs go out to the right. Because the timing cycle is fixed, we omit the timing inputs in these figures. For similar reasons, we also omit the reinforcement inputs.

In the simulations described here, the weights in the NATs start at zero making the network outputs completely random in the sense that on any given trial, all outputs are equally likely to occur, independent of past or present inputs. As learning proceeds, some or all the weights become large, so that the NAT-net's selection of outputs is strongly influenced by some or all of its inputs and internal connections. (Note that if the weights do not start at zero, they can be driven close to zero by using negative reinforcement.) In general, the optimum behavior toward which the network adapts is deterministic. However, because the probabilities are never identically equal to zero or one, we apply an arbitrary criterion and say that a NAT-net has learned the appropriate behavior when that criterion is satisfied. In real biological systems, we cannot know the weights or the exact probabilities governing the behavior of the individual adaptive elements. Therefore, it is appropriate to use a criterion based on observable behavior. For example, the criterion might be that the network selects the correct response (and continues to receive appropriate reinforcement) 25 times in a row.

Note that NAT-nets can adapt appropriately when the environment is not deliberately trying to make the them behave in a particular way. For example, the environment may provide inputs according to some (not necessarily deterministic) pattern and there may be some independent mechanism that determines whether the NAT-net is responding appropriately or not and provides the reinforcement accordingly. One paradigm for this situation is a game in which the NAT-net and the environment are players. The reinforcement scheme is simple: if, according to the rules of the game, the NAT-net wins a play (= trial) of the game, reinforcement is +1 , if it loses, −1.

For a NAT-net to adapt appropriately in this situation, the game must consist of a series of similar plays. If the game is competitive, the best strategy a given player has depends on how much information he has about the opponent and vice versa. If a player assumes that his opponent is all-knowing, then his best strategy is to minimize his maximum loss and this often means playing at random, or a least according to certain probabilities. If a player knows a lot about how his opponent plays, his best strategy may be to maximize gain. This often means playing according to some deterministic strategy.

The example networks described here are special cases of three types: pattern producing (combinational multiple-output) networks, pattern recognizing (combinational multiple-input, multiple-layered, few-output) networks, and game playing (sequential) networks. The associative properties of NATs and NAT-nets

are not emphasized here because they are analogous to the well known associative properties of other related models.

## A Class of Simple Pattern Producing Networks

A simple class of pattern producing networks consists of the single-layer type shown in Fig. 1. Each of $M$ NATs in such a network has no inputs, only an output. As a consequence, each has only one weight, $w_0$. The network is a simple, adaptive, information source.

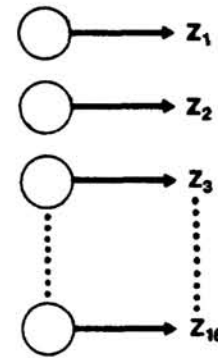

Consider first the case in which the network contains only one NAT and we wish to train it to always produce a simple "pattern," $+1$. We give positive reinforcement when it selects $+1$ and negative reinforcement otherwise. If $w_0$ starts at 0, it will quickly grow large making the probability of selecting $+1$ approach unity. The criterion we use for deciding that the network is trained is that it produce a string of 25 correct outputs. Table I

Fig. 1.   A Simple Pattern Producing Network

shows that in 100 sessions, this one-NAT network selected $+1$ output for the next 25 trials starting, on average, at trial 13.

Next consider a network with two NATs. They can produce four different output patterns. If both weights are 0, they will produce each of the patterns with equal probability. But they can be trained to produce one pattern (nearly) all the time. If we wish to train this subnetwork to produce the pattern (in vector notation) $[+1 \ +1]$, one strategy is to give no reinforcement if it produces patterns $[-1 \ +1]$ or $[+1 \ -1]$, give it positive reinforcement if it produces $[+1 \ +1]$ and negative reinforcement if it produces $[-1 \ -1]$. Table I shows that in 100 sessions, this network learned to produce the desired pattern (by producing a string of 25 correct outputs) in about 25 trials. Because we initially gave reinforcement only about 50% of the time, it took longer to train two NATS than one.

| M | Min | Ave | Max |
|---|-----|-----|-----|
| 1 | 1 | 13 | 26 |
| 2 | 8 | 25 | 43 |
| 4 | 18 | 35 | 60 |
| 8 | 44 | 70 | 109 |
| 16 | 49 | 115 | 215 |

Table I.   Training Times For Networks Per Fig. 1.

Next, consider the 16-NAT network in Fig. 1. Now there are $2^{16}$ possible patterns the network can produce. When all the weights are zero, each has probability $2^{-16}$ of being produced. An *ineffective* strategy for training this network is to provide positive reinforcement when the desired pattern is produced, negative reinforcement when its opposite is produced, and zero reinforcement otherwise. A better strategy is to focus on one output of the network at a time, training each NAT separately (as above) to have a high probability of producing the desired output. Once all are trained to a relatively high level, the network as a whole has a reasonable chance of producing exactly the correct output. Now we can provide positive reinforcement when it does and no reinforcement otherwise. With this two-stage hybrid strategy, the network will soon meet the training criterion. The time it takes to train a network of $M$ elements with a strategy of this type is roughly proportional to $M$, not $2^{(M-1)}$, as for the first strategy.

A still more efficient strategy is to alternate between a general substrategy and a substrategy focussed on keeping the network from getting "stuck." One effective general substrategy is to give positive reinforcement when more than half of the NATs select the desired output, negative reinforcement when less than half select the desired output, and no reinforcement when exactly half select the desired output. This substrategy starts out with approximately equal amounts of positive and negative reinforcement being applied. Soon, the network selects more than half of the outputs correctly more and more of the time. Unfortunately, there will tend to be a minority subset with low probability of selecting the correct output. At this stage, we must recognize this subset and switch to a substrategy that focuses on the elements of this subset following the strategy for one or two elements, above. When all NATs have a sufficiently high probability of selecting the desired output, training can conclude with the first substrategy.

The strategies used to obtain the results for $M = 4, 8$, and 16 in Table I were slightly more complicated variants of this two-part strategy. In all of them, a running average was kept of the number of right responses given by each NAT. Letting $C_i$ be the "correct" output for $z_i$, the running average after the $t^{th}$ trial, $A_i(t)$, is:

$$A_i(t) = BA_i(t-1) + C_i z_i(t) \qquad (3)$$

where $B$ is a fraction generally in the range 0.75 to 0.9. If $A_i(t)$ for a particular $i$ gets too far below the combined average for all $i$, then training focuses on the $i^{th}$ element until its average improves. The significance of the results given in Table I is not the details of the strategies used, nor how close the training times may be to the optimum. Rather, it is the demonstration that training strategies *exist* such that the training time grows significantly more slowly than in proportion to M.

### A Simple Pattern Recognizing Network

As mentioned above, there are fewer threshold logic functions of N variables (for $N > 1$) than the total possible functions. For $N = 2$, there are 14. The remining two are the "exclusive or" (XOR) and its complement. Multi-layered networks are needed to realize these functions, and an important test of any adaptive network model is its ability to learn XOR. The

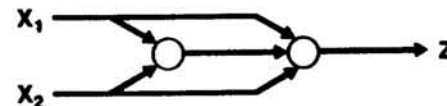

Fig. 2. A Two-Element Network That Learns XOR

network in Fig. 2 is one of the simplest networks capable of learning this function. Table II gives the results of 100 training sessions with this network. The strategy used to obtain these results again had two parts. The general part consisted of supplying each of the four possible input patterns to the network in rotation, giving appropriate reinforcement each trial. The second part involved

| Network | Function | Min | Ave | Max |
|---|---|---|---|---|
| Fig. 2 | OR | 18 | 57 | 106 |
| Fig. 2 | XOR | 218 | 681 | 1992 |
| Ref. 2 | XOR | ~700 | ~3500 | ~14,300 |
| Ref. 8 | XOR | 2232 | - | - |

Table II. Training Times For The Network In Fig. 2.

keeping a running average (similar to Eq. (3)) of the responses of the network by input combination. When the average for one combination fell significantly behind

the average for all, training was focused on just that combination until performance improved. The criterion used for deciding when training was complete was a sequence of 50 correct responses (for all input patterns together).

For comparison, Table II shows results for the same network trained to realize the normal OR function. Also shown for comparison are numbers taken from Refs. 2 and 8 for the equivalent network in those different models. These are nondeterministic and deterministic models, respectively. The numbers from Ref. 2 are not exactly comparable with the present results for several reasons. These include: 1) The criterion for judging when the task was learned was not the same; 2) In Ref. 2, the "wrong" reinforcement was deliberately applied 10% of the time to test learning in this situation; 3) Neither model was optimized for the particular task at hand. Nonetheless, if these (and other) differences were taken into account, it is likely that the NAT-net would have learned the XOR function significantly faster.

The significance of the present results is that they suggest that the use of a training strategy can not only prevent a network from getting stuck, but may also facilitate more rapid learning. Thus, such strategies can compensate, or more than compensate, for reduced complexity in the reinforcement algorithm.

### A Simple Game-Playing Network

Here, we consider NAT-nets in the context of the game of "matching pennies." In this game, each player has a stack of pennies. At each play of the game, each player places one of his pennies, heads up or heads down, but covered, in front of him. Each player uncovers his penny at the same time. If they match, player $A$ adds both to his stack, otherwise, player $B$ takes both.

Game theory says that the strategy of each player that minimizes his maximum loss is to play heads and tails at random. Then $A$ cannot predict $B$'s behavior and at best can win 50% of the time and likewise for $B$ with respect to $A$. This is a conservative strategy on the part of each player because each assumes that the other has (or can derive through a sequence of plays), and can use, information about the other player's strategy. Here, we make the different assumption that: 1) Player $B$ does not play at random, 2) Player $B$ has no information about $A$'s strategy, and 3) Player $B$ is incapable of inferring any information about $A$ through a sequence of plays and in any event is incapable of changing its strategy. Then, if $A$ has no information about $B$'s pattern of playing at the start of the game, $A$'s best course of action is to try to infer a non-random pattern in $B$'s playing through a sequence of plays and subsequently take advantage of that knowledge to win more often than 50% of the time. An adaptive NAT-net, as $A$, can adapt appropriately in situations of this type. For example, suppose a single NAT of the type in Fig. 1 plays $A$, where $+1$ output means heads, $-1$ output means tails. A third agent supplies reinforcement $+1$ if the NAT wins a play, $-1$ otherwise. Suppose $B$ plays heads with 0.55 probability and tails with 0.45 probability. Then $A$ will learn over time to play heads 100% of the time and thereby maximize its total winnings by winning 55% of the time.

A more complicated situation is the following. Suppose $B$ repeats its own move *two* plays ago 80% of the time, and plays the opposite 20% of the time. A NAT-net with the potential to adapt to this strategy and win 80% of the time is shown in Fig. 3. This is a sequential network shown in the standard form of a combinational network (in the dotted rectangle) plus a feedback path. The input to the network at time $t$ is $B$'s play at $t-1$. The output is $A$'s move. The top NAT selects its output at time $t$ based partly on the bottom NAT's output at time $t-1$. The bottom NAT selects its output at $t-1$ based on its input at that time which is $B$'s output at $t-2$. Thus, the network as a whole can learn to select its

output based on $B$'s play two time increments past. Simulation of 100 sessions resulted in the network learning to do this 98 times. On average, it took 468 plays (Min 20, max 4137) to reach the point at which the network repeated $B$'s move two times past on the next 50 plays. For two sessions the network got stuck (for an unknown number of plays greater than 25,000) playing the opposite of $B$'s last move or always playing tails. (The first two-part strategy found that trains the network to repeat $B$'s output two time increments past without getting stuck (not in the game-playing context) took an average of 260 trials (Min 25, Max 1943) to meet the training criterion.)

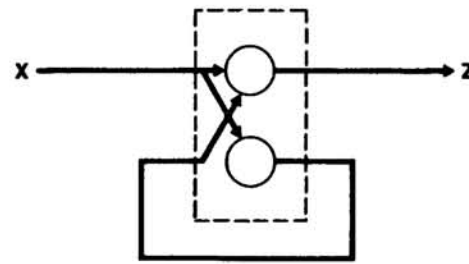

Fig. 3.   A Sequential Game-Playing Network

The significance of these results is that a *sequential* NAT-net can learn to produce appropriate behavior. Note that hidden NATs contributed to appropriate behavior for both this network and the one that learned XOR, above.

## CONCLUDING REMARKS

The examples above have been kept simple in order to make them readily understandable. They are not exhaustive in the sense of covering all possible types of situations in which NAT-nets can adapt appropriately. Nor are they definitive in the sense of proving generally and in what situations NAT-nets can adapt appropriately. Rather, they are illustrative in the sense of demonstrating a variety of significant adaptive abilities. They provide an existence proof that NAT-nets can adapt appropriately and relatively easily in a wide variety of situations.

The fact that *non*deterministic models can learn when the same reinforcement is applied to all adaptive elements, while deterministic models generally cannot, supports the hypothesis that animal nervous systems may be (partly) nondeterministic. Experimental characterization of how animal learning does, or does not get "stuck," as a function of learning environment or training strategy, would be a useful test of the ideas presented here.

## REFERENCES

1. Barto, A. G., "Game-Theoretic Cooperativity in Networks of Self-Interested Units," pp. 41-46 in Neural Networks for Computing, J. S. Denker, Ed., AIP Conference Proceedings 151, American Institute of Physics, New York, 1986.
2. Barto, A. G., Human Neurobiology, *4*, 229-256, 1985.
3. Barto, A. G., R. S. Sutton, and C. W. Anderson, IEEE Transactions on Systems, Man, and Cybernetics, SMC-13, No. 5, 834-846, 1983.
4. Barto, A. G., and P. Anandan, IEEE Transactions on Systems, Man, and Cybernetics, SMC-15, No. 3, 360-375, 1985.
5. Windecker, R. C., Information Sciences, *16*, 185-234 (1978).
6. Rumelhart, D. E., and J. L. McClelland, Parallel Distributed Processing, MIT Press, Cambridge, 1986.
7. Muroga, S., Threshold Logic And Its Applications, Wiley-Interscience, New York, 1971.
8. Rumelhart, D. E., G. E. Hinton, and R. J. Williams, Chapter 8 in Ref. 6.
